# Learning with Compressible Priors

**Volkan Cevher**
Rice University
`volkan@rice.edu`

## Abstract

We describe a set of probability distributions, dubbed *compressible priors*, whose independent and identically distributed (iid) realizations result in $p$-compressible signals. A signal $\boldsymbol{x} \in \mathbb{R}^N$ is called $p$-compressible with magnitude $R$ if its sorted coefficients exhibit a power-law decay as $|\boldsymbol{x}|_{(i)} \lesssim R \cdot i^{-d}$, where the decay rate $d$ is equal to $1/p$. $p$-compressible signals live close to $K$-sparse signals ($K \ll N$) in the $\ell_r$-norm ($r > p$) since their best $K$-sparse approximation error decreases with $\mathcal{O}\left(R \cdot K^{1/r-1/p}\right)$. We show that the membership of generalized Pareto, Student's $t$, log-normal, Fréchet, and log-logistic distributions to the set of compressible priors depends only on the distribution parameters and is independent of $N$. In contrast, we demonstrate that the membership of the generalized Gaussian distribution (GGD) depends both on the signal dimension and the GGD parameters: the expected decay rate of $N$-sample iid realizations from the GGD with the shape parameter $q$ is given by $1/\left[q \log\left(N/q\right)\right]$. As stylized examples, we show via experiments that the wavelet coefficients of natural images are $1.67$-compressible whereas their pixel gradients are $0.95 \log\left(N/0.95\right)$-compressible, on the average. We also leverage the connections between compressible priors and sparse signals to develop new iterative re-weighted sparse signal recovery algorithms that outperform the standard $\ell_1$-norm minimization. Finally, we describe how to learn the hyperparameters of compressible priors in underdetermined regression problems by exploiting the geometry of their order statistics during signal recovery.

## 1 Introduction

Many problems in signal processing, machine learning, and communications can be cast as a linear regression problem where an unknown signal $\boldsymbol{x} \in \mathbb{R}^N$ is related to its observations $\boldsymbol{y} \in \mathbb{R}^M$ via

$$\boldsymbol{y} = \boldsymbol{\Phi}\boldsymbol{x} + \boldsymbol{n}. \tag{1}$$

In (1), the observation matrix $\boldsymbol{\Phi} \in \mathbb{R}^{M \times N}$ is a non-adaptive measurement matrix with random entries in compressive sensing (CS), an over-complete dictionary of features in sparse Bayesian learning (SBL), or a code matrix in communications [1, 2]. The vector $\boldsymbol{n} \in \mathbb{R}^M$ usually accounts for physical noise with partially or fully known distribution, or it models bounded perturbations in the measurement matrix or the signal.

Because of its theoretical and practical interest, we focus on the instances of (1) where there are more unknowns than equations, i.e., $M < N$. Hence, determining $\boldsymbol{x}$ from $\boldsymbol{y}$ in (1) is ill-posed: $\forall \boldsymbol{v} \in$ kernel $(\boldsymbol{\Phi})$, $\boldsymbol{x} + \boldsymbol{v}$ defines a solution space that produces the same observations $\boldsymbol{y}$. Prior information is therefore necessary to distinguish the true $\boldsymbol{x}$ among the infinitely many possible solutions. For instance, CS and SBL frameworks assume that the signal $\boldsymbol{x}$ belongs to the set of sparse signals. By sparse, we mean that at most $K$ out of the $N$ signal coefficients are nonzero where $K \ll N$. CS and SLB algorithms then regularize the solution space by signal priors that promote sparseness and they have been extremely successful in practice in a number of applications even if $M \ll N$ [1–3].

Unfortunately, prior information by itself is not sufficient to recover $\boldsymbol{x}$ from noisy $\boldsymbol{y}$. Two more key ingredients are required: (*i*) the observation matrix $\boldsymbol{\Phi}$ must stably *embed* (or encode) the set of signals $\boldsymbol{x}$ into the space of $\boldsymbol{y}$, and (*ii*) a tractable *decoding* algorithm must exist to map $\boldsymbol{y}$ back to $\boldsymbol{x}$. By stable embedding, we mean that $\boldsymbol{\Phi}$ is bi-Lipschitz where the encoding $\boldsymbol{x} \rightarrow \boldsymbol{\Phi}\boldsymbol{x}$ is one to one and the inverse mapping $\Delta = \{\Delta\left(\boldsymbol{\Phi}\boldsymbol{x}\right) \rightarrow \boldsymbol{x}\}$ is smooth. The bi-Lipschitz property of $\boldsymbol{\Phi}$ is crucial to ensure the stability in decoding $\boldsymbol{x}$ by controlling the amount by which perturbations of the

observations are amplified [1, 4]. Tractable decoding is important for practical reasons as we have limited time and resources, and it can clearly restrict the class of usable signal priors.

In this paper, we describe *compressible prior distributions* whose independent and identically distributed (iid) realizations result in compressible signals. A signal is compressible when sorted magnitudes of its coefficients exhibit a power-law decay. For certain decay rates, compressible signals live close to the sparse signals, i.e., they can be well-approximated by sparse signals. It is well-known that the set of $K$-sparse signals has stable and tractable encoder-decoder pairs $(\mathbf{\Phi}, \Delta)$ for $M$ as small as $\mathcal{O}(K \log (N/K))$ [1, 5]. Hence, an $N$-dimensional compressible signal with the proper decay rate inherits the encoder-decoder pairs of its $K$-sparse approximation for a given approximation error, and can be stably embedded into dimensions logarithmic in $N$.

Compressible priors analytically summarize the set of compressible signals and shed new light on underdetermined linear regression problems by building upon the literature on sparse signal recovery. Our main results are summarized as follows:

1) By using order statistics, we show that the compressibility of the iid realizations of generalized Pareto, Student's $t$, Fréchet, and log-logistics distributions is independent of the signals' dimension. These distributions are natural members of compressible priors: they truly support logarithmic dimensionality reduction and have important parameter learning guarantees from finite sample sizes. We demonstrate that probabilistic models for the wavelet coefficients of natural images must also be a natural member of compressible priors.

2) We point out a common misconception about the generalized Gaussian distribution (GGD): GGD generates signals that lose their compressibility as $N$ grows. For instance, special cases of the GGD distribution, e.g., Laplacian distribution, are commonly used as sparsity promoting priors in CS and SBL problems where $M$ is assumed to grow logarithmically with $N$ [1–3, 6]. We show that signals generated from Laplacian distribution can only be stably embedded into lower dimensions that grow proportional to $N$. Hence, we identify an inconsistency between the decoding algorithms motivated by the GGD distribution and their sparse solutions.

3) We use compressible priors as a scaffold to build new decoding algorithms based on Bayesian inference arguments. The objective of these algorithms is to approximate the signal realization from a compressible prior as opposed to pragmatically producing sparse solutions. Some of these new algorithms are variants of the popular iterative re-weighting schemes [3, 6–8]. We show how the tuning of these algorithms explicitly depends on the compressible prior parameters, and how to learn the parameters of the signal's compressible prior on the fly while recovering the signal.

The paper is organized as follows. Section 2 provides the necessary background on sparse signal recovery. Section 3 mathematically describes the compressible signals and ties them with the order statistics of distributions to introduce compressible priors. Section 4 defines compressible priors, identifies common misconceptions about the GGD distribution, and examines natural images as instances of compressible priors. Section 5 derives new decoding algorithms for underdetermined linear regression problems. Section 6 describes an algorithm for learning the parameters of compressible priors. Section 7 provides simulations results and is followed by our conclusions.

## 2 Background on Sparse Signals

Any signal $\boldsymbol{x} \in \mathbb{R}^N$ can be represented in terms of $N$ coefficients $\boldsymbol{\alpha}_{N \times 1}$ in a basis $\boldsymbol{\Psi}_{N \times N}$ via $\boldsymbol{x} = \boldsymbol{\Psi}\boldsymbol{\alpha}$. Signal $\boldsymbol{x}$ has a *sparse representation* if only $K \ll N$ entries of $\boldsymbol{\alpha}$ are nonzero. To account for sparse signals in an appropriate basis, (1) should be modified as $\boldsymbol{y} = \boldsymbol{\Phi}\boldsymbol{x} + \boldsymbol{n} = \boldsymbol{\Phi}\boldsymbol{\Psi}\boldsymbol{\alpha} + \boldsymbol{n}$.

Let $\Sigma_K$ denote the set of all $K$-sparse signals. When $\boldsymbol{\Phi}$ in (1) satisfies the so-called restricted isometry property (RIP), it can be shown that $\boldsymbol{\Phi}\boldsymbol{\Psi}$ defines a bi-Lipschitz embedding of $\Sigma_K$ into $R^M$ [1, 4, 5]. Moreover, RIP implies the recovery of $K$-sparse signals to within a given error bound, and the best attainable lower bounds for $M$ are related to the Gelfand width of $\Sigma_K$, which is logarithmic in the signal dimension, i.e., $M = \mathcal{O}(K \log (N/K))$ [5]. Without loss of generality, we restrict our attention in the sequel to canonically sparse signals and assume that $\boldsymbol{\Psi} = \boldsymbol{I}$ (the $N \times N$ identity matrix) so that $\boldsymbol{x} = \boldsymbol{\alpha}$.

With the sparsity prior and RIP assumptions, inverse maps can be obtained by solving the following convex problems:

$$\Delta_1(\boldsymbol{y}) = \arg \min \|\boldsymbol{x}'\|_1 \text{ s.t. } \boldsymbol{y} = \boldsymbol{\Phi}\boldsymbol{x}',$$
$$\Delta_2(\boldsymbol{y}) = \arg \min \|\boldsymbol{x}'\|_1 \text{ s.t. } \|\boldsymbol{y} - \boldsymbol{\Phi}\boldsymbol{x}'\|_2 \le \epsilon, \tag{2}$$
$$\Delta_3(\boldsymbol{y}) = \arg \min \|\boldsymbol{x}'\|_1 + \tau\|\boldsymbol{y} - \boldsymbol{\Phi}\boldsymbol{x}'\|_2^2,$$

where $\epsilon$ and $\tau$ are constants, and $\|\boldsymbol{x}\|_r \triangleq \left(\sum_i |x_i|^r\right)^{1/r}$. The decoders $\Delta_i$ ($i = 1, 2$) are known as basis pursuit (BP) and basis pursuit denoising (BPDN), respectively; and, $\Delta_3$ is a scalarization of BPDN [1, 9]. They also have the following deterministic worst-case guarantee when $\boldsymbol{\Phi}$ has RIP:

$$\|\boldsymbol{x} - \Delta(\boldsymbol{y})\|_2 \le C_1 \frac{\|\boldsymbol{x} - \boldsymbol{x}_K\|_1}{\sqrt{K}} + C_2 \|n\|_2, \tag{3}$$

where $C_{1,2}$ are constants, $\boldsymbol{x}_K$ is the best $K$-term approximation, i.e., $\boldsymbol{x}_K = \arg\min_{\|\boldsymbol{x}'\|_0 \le K} \|\boldsymbol{x} - \boldsymbol{x}'\|_r$ for $r \ge 1$, and $\|\boldsymbol{x}\|_0$ is a pseudo-norm that counts the number of nonzeros of $\boldsymbol{x}$ [1, 4, 5]. Note that the error guarantee (3) is adaptive to each given signal $\boldsymbol{x}$ because of the definition of $\boldsymbol{x}_K$. Moreover, the guarantee does not assume that the signal is sparse.

# 3 Compressible Signals, Order Statistics and Quantile Approximations

We define a signal $\boldsymbol{x}$ as *p-compressible* if it lives close to the shell of the weak-$\ell_p$ ball of radius $R$ ($sw\ell_p(R)$–pronounced as *swell p*). Defining $\bar{x}_i = |x_i|$, we arrange the signal coefficients $x_i$ in decreasing order of magnitude as

$$\bar{x}_{(1)} \ge \bar{x}_{(2)} \ge \ldots \ge \bar{x}_{(N)}. \tag{4}$$

Then, when $\boldsymbol{x} \in sw\ell_p(R)$, the $i$-th ordered entry $\bar{x}_{(i)}$ in (4) obeys

$$\bar{x}_{(i)} \lesssim R \cdot i^{-1/p}, \tag{5}$$

where $\lesssim$ means "less than or approximately equal to." We deliberately substitute $\lesssim$ for $\le$ in the $p$-compressibility definition of [1] to reduce the ambiguity of multiple feasible $R$ and $p$ values. In Section 6, we describe a geometric approach to learn $R$ and $p$ so that $R \cdot i^{-1/p} \approx \bar{x}_{(i)}$.

Signals in $sw\ell_p(R)$ can be well-approximated by sparse signals as the best $K$-term approximation error decays rapidly to zero as

$$\|\boldsymbol{x} - \boldsymbol{x}_K\|_r \lesssim (r/p - 1)^{-1/r} R K^{1/r - 1/p}, \text{ when } p < r. \tag{6}$$

Given $M$, a good rule of thumb is to set $K = M/[C \log(N/M)]$ ($C \approx 4$ or $5$) and use (6) to predict the approximation error for the decoders $\Delta_i$ in Section 2. Since the decoding guarantees are bounded by the best $K$-term approximation error in $\ell_1$ (i.e., $r = 1$; cf. (3)), we will restrict our attention to $\boldsymbol{x} \in sw\ell_p$ where $p < 1$. Including $p = 1$ adds a logarithmic error factor to the approximation errors, which is not severe; however, it is not considered in this paper to avoid a messy discussion.

Suppose now the individual entries $x_i$ of the signal $\boldsymbol{x}$ are random variables (RV) drawn iid with respect to a probability density function (pdf) $f(x)$, i.e., $x_i \sim f(x)$ for $i = 1, \ldots, N$. Then, $\bar{x}_{(i)}$'s in (4) are also RV's and are known as the *order statistics* (OS) of yet another pdf $\bar{f}(\bar{x})$, which can be related to $f(x)$ in a straightforward manner: $\bar{f}(\bar{x}) = f(\bar{x}) + f(-\bar{x})$. Note that even though the RV's $x_i$ (hence, $\bar{x}_i$) are iid, the RV's $\bar{x}_{(i)}$ are statistically dependent.

The concept of OS enables us to create a link between signals summarized by pdf's and their compressibility, which is a deterministic property after the signals are realized. The key to establishing this link turns out to be the parameterized form of the quantile function of the pdf $\bar{f}(\bar{x})$. Let $\bar{F}(\bar{x}) = \int_0^{\bar{x}} \bar{f}(v) dv$ be the cumulative distribution function (CDF) and $u = \bar{F}(\bar{x})$. The quantile function $\bar{F}^\star(u)$ of $\bar{f}(\bar{x})$ is then given by the inverse of its CDF: $\bar{F}^\star(u) = \bar{F}^{-1}(u)$. We will refer to $\bar{F}^\star(u)$ as the magnitude quantile function (MQF) of $f(x)$.

A well-known *quantile approximation* to the expected OS of a pdf is given by [10]:

$$E[\bar{x}_{(i)}] = \bar{F}^\star \left(1 - \frac{i}{N+1}\right), \tag{7}$$

where $E[\cdot]$ is the expected value. Moreover, we have the following moment matching approximation

$$\bar{x}_{(i)} \sim \mathcal{N} \left(E[\bar{x}_{(i)}], \frac{\frac{i}{N}\left(1 - \frac{i}{N}\right)}{N \left[f\left(E[\bar{x}_{(i)}]\right)\right]^2}\right), \tag{8}$$

which can be used to quantify how much the actual realizations $\bar{x}_{(i)}$ deviate from $E[\bar{x}_{(i)}]$. For instance, these deviations for $i > K$ can be used to bound the statistical variations of the best $K$-term approximation error. In practice, the deviations are relatively small for compressible priors. In Sections 4–6, we will use the quantile approximation in (7) as our basis to motivate the set of compressible priors, derive recovery algorithms for $\boldsymbol{x}$, and learn the parameters of compressible priors during recovery.

Table 1: Example distributions and the $sw\ell_p(R)$ parameters of their iid realizations

| Distribution | pdf | $R$ | $p$ |
|---|---|---|---|
| Generalized Pareto | $\frac{q}{2\lambda}\left(1+\frac{|x|}{\lambda}\right)^{-(q+1)}$ | $\lambda N^{1/q}$ | $q$ |
| Student's $t$ | $\frac{\Gamma((q+1)/2)}{\sqrt{2\pi}\lambda\Gamma(q/2)}\left(1+\frac{x^2}{\lambda^2}\right)^{-(q+1)/2}$ | $\left[\frac{2\Gamma((q+1)/2)}{\sqrt{\pi}q\Gamma(q/2)}\right]^{1/q}\lambda N^{1/q}$ | $q$ |
| Fréchet | $(q/\lambda)\,(x/\lambda)^{-(q+1)}\,\mathrm{e}^{-(x/\lambda)^{-q}}$ | $\lambda N^{1/q}$ | $q$ |
| Log-Logistic | $\frac{(q/\lambda)(x/\lambda)^{q-1}}{[1+(x/\lambda)^q]^2}$ | $\lambda N^{1/q}$ | $q$ |
| Generalized Gaussian | $\frac{q}{2\lambda\Gamma(1/q)}\,\mathrm{e}^{-(|x|/\lambda)^q}$ | $\lambda\max\{1,\Gamma(1+1/q)\}\log^{1/q}(N/q)$ | $q\log(N/q)$ |
| Weibull | $(q/\lambda)\,(x/\lambda)^{q-1}\,\mathrm{e}^{-(x/\lambda)^q}$ | $\lambda\log^{1/q}N$ | $q\log N$ |
| Gamma | $\frac{1}{\lambda\Gamma(q)}\,(x/\lambda)^{q-1}\,\mathrm{e}^{-x/\lambda}$ | $\lambda\max\{1,\Gamma(1+1/q)^q\}\log(qN)$ | $\log(qN)$ |
| Log-Normal | $\frac{q}{\sqrt{2\pi}x}\mathrm{e}^{-(q\log(x/\lambda))^2/2}$ | $\lambda\mathrm{e}^{\sqrt{2\log N}/q}$ | $\sqrt{2\log N}q$ |

## 4 Compressible Priors

A *compressible prior* $f(x;\boldsymbol{\theta})$ in $\ell_r$ is a pdf with parameters $\boldsymbol{\theta}$ whose MQF satisfies

$$\bar{F}^\star\left(1-\frac{i}{N+1}\right)\lesssim R(N,\boldsymbol{\theta})\cdot i^{-1/p(N,\boldsymbol{\theta})},\text{ where }R>0\text{ and }p<r. \qquad (9)$$

Table 4 lists example pdf's, parameterized by $\boldsymbol{\theta}=(q,\lambda)\succ 0$, and the $sw\ell_p(R)$ parameters of their $N$-sample iid realizations. In this paper, we fix $r=1$ (cf. Section 3); hence, the example pdf's are compressible priors whenever $p<1$. In (9), we make it explicit that the $sw\ell_p(R)$ parameters can depend on the parameters $\boldsymbol{\theta}$ of the specific compressible prior as well as the signal dimension $N$. The dependence of the parameter $p$ on $N$ is of particular interest since it has important implications in signal recovery as well as parameter learning from finite sample sizes, as discussed below.

We define *natural $p$-compressible priors* as the set $\mathfrak{N}_p$ of compressible priors such that $p=p(\boldsymbol{\theta})<1$ is independent of $N$, $\forall f(x;\boldsymbol{\theta})\in\mathfrak{N}_p$. It is possible to prove that we can capture most of the $\ell_1$-energy in an $N$-sample iid realization from a natural $p$-compressible prior by using a constant $K$, i.e., $\|\boldsymbol{x}-\boldsymbol{x}_K\|_1\le\epsilon\|\boldsymbol{x}\|_1$ for any desired $0<\epsilon\ll 1$ by choosing $K=\lceil(p/\epsilon)^{\frac{p}{1-p}}\rceil$. Hence, $N$-sample iid signal realizations from the compressible priors in $\mathfrak{N}_p$ can be truly embedded into dimensions $M$ that grow logarithmically with $N$ with tractable decoding guarantees due to (3). $\mathfrak{N}_p$ members include the generalized Pareto (GPD), Fréchet (FD), and log-logistic distributions (LLD).

It then only comes as a surprise that generalized Gaussian distribution (GGD) is not a natural $p$-compressible prior since its iid realizations lose their compressibility as $N$ grows (cf. Table 4). While it is common practice to use a GGD prior with $q\le 1$ for sparse signal recovery, we have no recovery guarantees for signals generated from GGD when $M$ grows logarithmically with $N$ in (1).[1] In fact, to be $p$-compressible, the shape parameter of a GGD prior should satisfy $q=N\mathrm{e}^{W_{-1}(-p/N)}$, where $W_{-1}(\cdot)$ is the Lambert $W$-function with the alternate branch. As a result, the learned GGD parameters from dimensionality-reduced data will in general depend on the dimension and may not generalize to other dimensions. Along with GGD, Table 4 shows how Weibull, gamma, and log-normal distributions are dimension-restricted in their membership to the set of compressible priors.

Wavelet coefficients of natural images provide a stylized example to demonstrate why we should care about the dimensional independence of the parameter $p$.[2] As a brief background, we first note that research in natural image modeling to date has had two distinct approaches, with one focusing on deterministic explanations and the other pursuing probabilistic models [12]. Deterministic approaches operate under the assumption that the natural images belong to Besov spaces, having a bounded number of derivatives between edges. Unsurprisingly, wavelet thresholding is proven near-optimal for representing and denoising Besov space images. As the simplest example, the magnitude sorted discrete wavelet coefficients $\bar{w}_{(i)}$ of a Besov $q$-image should satisfy $\bar{w}_{(i)}=R\cdot i^{-1/q}$. The probabilistic approaches, on the other hand, exploit the power-law decay of the power spectra of images and fit various pdf's, such as GGD and the Gaussian scale mixtures, to the histograms of wavelet

coefficients while trying to simultaneously capture the dependencies observed in the marginal and joint distributions of natural image wavelet coefficients. Probabilistic approaches are quite important in image compression because optimal compressors quantize the wavelet coefficients according to the estimated distributions, dictating the image compression limits via Shannon's coding theorem.

We conjecture that probabilistic models that summarize the wavelet coefficients of natural images belong to the set of natural (non-iid) $p$-compressible priors. We base our claim on two observations: 1) Due to the multiscale nature of the wavelet transform, the decay profile of the magnitude sorted wavelet coefficients are scale-invariant, i.e., preserved at different resolutions, where lower resolutions inherit the highest resolution. Hence, probabilistic models that explain the wavelet transform of *any signals* should exhibit this decay profile inheritance property. 2) The magnitude sorted wavelet coefficients of natural images exhibit a constant decay rate, as expected of Besov space images. Section 7.2 demonstrates the ideas using natural images from the Berkeley natural images database.

# 5 Signal Decoding Algorithms

Convex problems to recover sparse or compressible signals in (2) are usually motivated by Bayesian inference. In a similar fashion, we formalize two new decoding algorithms below by assuming prior distributions on the signal $x$ and the noise $n$, and then asking inference questions given $y$ in (1).

## 5.1 Fixed point continuation for a non-iid compressible prior

The multivariate Lomax distribution (MLD) provides an elementary example of a non-iid compressible prior. The pdf of the distribution is given by $\mathrm{MLD}(\boldsymbol{x}; q, \boldsymbol{\lambda}) \propto \left(1 + \sum_{i=1}^{N} \lambda_i^{-1} |x_i|\right)^{-q-N}$ [13]. For MLD, the marginal distribution of the signal coefficients is GPD, i.e., $x_i \sim \mathrm{GPD}(x; q, \lambda_i)$. Moreover, given $n$-realizations $\boldsymbol{x}_{1:n}$ of MLD ($n \leq N$), the joint marginal distribution of $\boldsymbol{x}_{n+1:N}$ is $\mathrm{MLD}(\boldsymbol{x}_{n+1:N}; q+k, \boldsymbol{\lambda}_{n+1:N} \left(1 + \sum_{i=1}^{n} \lambda_i^{-1} |x_i|\right)^{-1})$. In the sequel, we assume $\lambda_i = \lambda \; \forall i$, for which it can be proved that MLD is compressible with $p = 1$ [14]. For now, we will only demonstrate this property via simulations in Section 7.1. With the MLD prior on $x$, we focus on only two optimization problems below, one based on BP and the other based on maximum a posteriori (MAP) estimation. Other convex formulations, such as BPDN ($\Delta_2$ in (2)) and LASSO [15], trivially follow.

1) *BP Decoder:* When there is no noise, the observations are given by $\boldsymbol{y} = \boldsymbol{\Phi x}$, which has infinitely many solutions, as discussed in Section 1. In this case, we can exploit the MLD likelihood function to regularize the solution space. For instance, when we ask for the solution that maximizes the MLD likelihood given $y$, it is easy to see that we obtain the BP decoder formulation, i.e., $\Delta_1(\boldsymbol{y})$ in (2).

2) *MAP Decoder:* Suppose that the noise coefficients ($n_i$'s in (1)) are iid Gaussian with zero mean and variance $\sigma^2$, $n_i \sim \mathcal{N}(n; 0, \sigma^2)$. Although many inference questions are possible, here we seek the mode of the posterior distribution to obtain a point estimate, also known as the MAP estimate. Since we have $f(\boldsymbol{y}|\boldsymbol{x}) = \mathcal{N}(\boldsymbol{y} - \boldsymbol{\Phi x}; 0, \sigma^2 \boldsymbol{I}_{M \times M})$ and $f(\boldsymbol{x}) = \mathrm{MLD}(\boldsymbol{x}; q, \lambda)$, the MAP estimate can be derived using the Bayes rule as $\widehat{\boldsymbol{x}}_{\mathrm{MAP}} = \arg\max_{\boldsymbol{x}'} f(\boldsymbol{y}|\boldsymbol{x}') f(\boldsymbol{x}')$, which is explicitly given by

$$\widehat{\boldsymbol{x}}_{\mathrm{MAP}} = \arg\min_{\boldsymbol{x}'} \|\boldsymbol{y} - \boldsymbol{\Phi x}'\|_2^2 + 2\sigma^2(q+N)\log\left(1 + \lambda^{-1}\|\boldsymbol{x}'\|_1\right). \tag{10}$$

Unfortunately, we stumble upon a non-convex problem in (10) during our quest for the MAP estimate. We circumvent the non-convexity in (10) using a *majorization-minimization* idea where we iteratively obtain a tractable upperbound on the log-term in (10) using the following inequality: $\forall u, v \in (0, \infty), \; \log u \leq \log v + u/v - 1$. After some straightforward calculus, we obtain the iterative decoder below, indexed by $k$, where $\widehat{\boldsymbol{x}}_{\{k\}}$ is the $k$-th iteration estimate ($\widehat{\boldsymbol{x}}_{\{0\}} = 0$):

$$\widehat{\boldsymbol{x}}_{\{k\}} = \arg\min_{\boldsymbol{x}'} \|\boldsymbol{y} - \boldsymbol{\Phi x}'\|_2^2 + \nu_k \|\boldsymbol{x}'\|_1, \text{ where } \nu_k = \frac{2\sigma^2(q+N)}{\lambda + \|\widehat{\boldsymbol{x}}_{\{k-1\}}\|_1}. \tag{11}$$

The decoding approach in (11) can be viewed as a continuation (or a homotopy) algorithm where a fixed point is obtained at each iteration, similar to [16]. This decoding scheme has provable, linear convergence guarantees when $\|\widehat{\boldsymbol{x}}_{\{k\}}\|_1$ is strictly increasing $\Uparrow$ (equivalently, $\nu_k \Downarrow$) [16].

## 5.2 Iterative $\ell_s$-decoding for iid scale mixtures of GGD

We consider a generalization of GPD and the Student's $t$ distribution, which we will denote as the generalized Gaussian gamma scale mixture distribution (SMD, in short), whose pdf is given by $\mathrm{SMD}(x; q, \lambda, s) \propto (1 + |x|^s / \lambda^s)^{-(q+1)/s}$. The additional parameter $s$ of SMD modulates its OS near the origin. It can be proved that SMD is $p$-compressible with $p = q$ [14]. SMD, for instance, arises through the following interaction of the gamma distribution and GGD: $x = a^{-1/s} b$, $a \sim \mathrm{Gamma}(a; q/s, \lambda^{-s})$, and $b \sim \mathrm{GGD}(b; s, 1)$. Given $a$, the distribution of $x$ is a scaled GGD:

$f(x|a) \sim \mathrm{GGD}(x; s, a^{-1})$. Marginalizing $a$ from $f(x|a)$, we reach the SMD as the true underlying distribution of $x$. SMD arise in multiple contexts, such as the SLB framework that exploit Student's $t$ (i.e., $s = 2$) for learning problems [2], and the Laplacian and Gaussian scale mixtures (i.e., $s = 1$ and 2, respectively) that model natural images [17, 18].

Due to lack of space, we only focus on noiseless observations in (1). We assume that $x$ is an $N$-sample iid realization from $\mathrm{SMD}(x; q, \lambda, s)$ with known parameters $(q, \lambda, s) \succ 0$ and choose a solution $\widehat{x}$ that maximizes the SMD likelihood to find the true vector $x$ among the kernel of $\Phi$:

$$\widehat{x} = \max_{x'} \mathrm{SMD}(x; q, \lambda, s) = \min_{x'} \sum_i \log \left( 1 + \lambda^{-s} |x_i|^s \right), \text{ s.t. } y = \Phi x'. \tag{12}$$

The majorization-minimization trick in Section 5.1 also circumvents the non-convexity in (12):

$$\widehat{x}_{\{k\}} = \min_{x'} \sum_i w_{i,\{k\}} |x_i|^s, \text{ s.t. } y = \Phi x'; \text{ where } w_{i,\{k\}} = \left( \lambda^s + \left| x_{i,\{k\}} \right|^s \right)^{-1}. \tag{13}$$

The decoding scheme in (13) is well-known as the iterative re-weighted $\ell_s$ algorithms [7, 19–21].

## 6    Parameter Learning for Compressible Distributions

While deriving decoding algorithms in Section 5, we assumed that the signal coefficients $x_i$ are generated from a compressible prior $f(x; \theta)$ and that $\theta$ is known. We now relax the latter assumption and discuss how to simultaneously estimate $x$ and learn the parameters $\theta$.

When we visualize the joint estimation of $x$ and $\theta$ from $y$ in (1) as a graphical model, we immediately realize that $x$ creates a Markov blanket for $\theta$. Hence, to determine $\theta$, we have to estimate the signal coefficients. When $\Phi$ has the stable embedding property, we know that the decoding algorithms can obtain $x$ with approximation guarantees, such as (3). Then, given $x$, we can choose an estimator for $\theta$ via standard Bayesian inference arguments. Unfortunately, this argument leads to one important road block: estimation of the signal $x$ without knowing the prior parameters $\theta$.

A naïve approach to overcoming this road block is to split the optimization space and alternate on $x$ and $\theta$ while optimizing the Bayesian objective. Unfortunately, there is one important and unrecognized bug in this argument: the estimated signal values are in general not iid, hence we would be minimizing the wrong Bayesian objective to determine $\theta$. To see this, we first note that the recovered signals $\widehat{x}$ in general consist of $M \ll N$ non-zero coefficients that mimic the best $K$-term approximation of the signal $x_K$ and some other coefficients that explain the small tail energy. We then recall from Section (3) that the coefficients of $x_K$ are statistically dependent. Hence, at least partially, the significant coefficients of $\widehat{x}$ are also dependent. One way to overcome this dependency issue is to treat the recovered signals as if they are drawn iid from a censored GPD. However, the optimization becomes complicated and the approach does not provide any additional guarantees.

As an alternative, we propose to exploit geometry and use the consensus among the coefficients in fitting the $sw\ell_p(R)$ parameters via the auxiliary signal estimates $\widehat{x}_{\{k\}}$ during iterative recovery. To do this, we employ Fischler and Bolles' probabilistic random sampling consensus (RANSAC) algorithm [22] to fit a line, whose $y$-intercept is $\log R(N, \theta)$ and whose slope is $1/p(N, \theta)$:

$$\log \left| \widehat{x}_{i,\{k\}} \right| = \log R(N, \theta) - \frac{1}{p(N, \theta)} \log i, \text{ for } i = 1, \dots, K; \text{ where } K = M/[C \log(N/M)], \tag{14}$$

where $C \approx 4, 5$ as discussed in Section. 3. RANSAC provides excellent results with high probability even if the data contains significant outliers. Because of its probabilistic nature, it is computationally efficient. The RANSAC algorithm requires a threshold to gate the observations and count how much a proposed solution is supported by the observations [22]. We determine this threshold by bounding the tail probability that the OS of a compressible prior will be out of bounds. For the pseudo-code and further details of the RANSAC algorithm, cf. [22].

## 7    Experiments

### 7.1    Order Statistics

To demonstrate the $sw\ell_p(R)$ decay profile of $p$-compressible priors, we generated iid realizations of GGD with $q = 1$ (LD) and GPD with $q = 1$, and (non-iid) realizations of MLD with $q = 1$ of varying signal dimensions $N = 10^j$, where $j = 2, 3, 4, 5$. We sorted the magnitudes of the signal coefficients, normalized them by their corresponding value of $R$. We then plotted the results on a log-log scale in Fig. 1. At *http://dsp.rice.edu/randcs*, we provide a MATLAB routine (*randcs.m*) so that it is easy to repeat the same experiment for the rest of the distributions in Table 4.

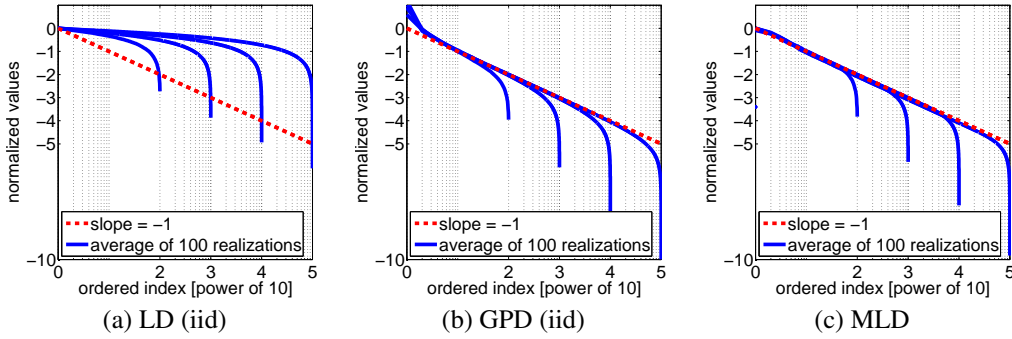

Figure 1: *Numerical illustration of the $sw\ell_p(R)$ decay profile of three different pdfs.*

To live in $sw\ell_p(1)$ with $0 < p \le 1$, the slope of the resulting curve must be less than or equal to $-1$. Figure 1(a) illustrates that the iid LD slope is much greater than $-1$ and moreover logarithmically grows with $N$. In contrast, Fig. 1(b) shows that iid GPD with $q = 1$ exhibits the constant slope of $-1$ that is independent of $N$. MLD with $q = 1$ also delivers such a slope (Fig. 1(c)). The latter two distributions thus produce compressible signal realizations, while the Laplacian does not.

## 7.2 Natural Images

We investigate the images from the Berkeley natural images database in the context of $p$-compressible priors. We randomly sample 100 image patches of varying sizes $N = 2^j \times 2^j$ ($j = 3, \ldots, 8$), take their wavelet transforms (scaling filter: daub2), and plot the average of their magnitude ordered wavelet coefficients in Figs. 2(a) and (b) (solid lines). Figure 2(c) also illustrates the OS of the pixel gradients, which are of particular interest in many applications.

Along with the wavelet coefficients, Fig. 2(a) superposes the expected OS of GPD with $q = 1.67$ and $\lambda = 10$ (dashed line), given by $\bar{x}_{(i)}\{\text{GPD}(q, \lambda)\} = \lambda \left[ (N+1)^{1/q} i^{-1/q} - 1 \right]$ ($i = 1, \ldots, N$). Although wavelet coefficients of natural images do not follow an iid distribution, they exhibit a constant decay rate, which can be well-approximated by an iid GPD distribution. This apparent constant decay rate is well-explained by the decay profile inheritance of the wavelet transform across different resolutions and supports the Besov space assumption used in the deterministic approaches. The GPD rate of $q = 1.67$ implies a disappointing $\mathcal{O}(K^{-0.1})$ approximation rate in the $\ell_2$-norm vs. the theoretical $\mathcal{O}(K^{-0.5})$ rate [23]. Moreover, we lose all the guarantees in the $\ell_1$-norm.

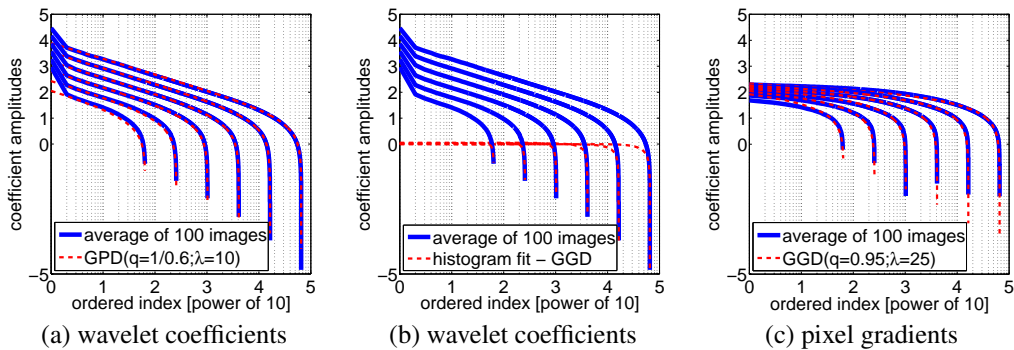

Figure 2: *Approximation of the order statistics and histograms of natural images with GPD and GGD.*

In contrast, Fig. 2(b) demonstrates the GGD histogram fits to the wavelet coefficients, where the GGD exponent $q \in [0.5, 1]$ depends on the particular dimension and decreases as $N$ increases. The histogram matching is common practice in the existing probabilistic approaches (e.g., [18]) to determine pdf's that explain the statistics of natural images. Typically, least square error metrics or Kullback-Liebler (KL) divergence measures are used. Although the GGD fit via histogram matching in Fig. 2(b) deceptively appears to fit a small number of coefficients, we emphasize the log-log scale of the plots and mention that there is a significant number of coefficients in the narrow space where the GGD distribution is a good fit. Unfortunately, these approaches approximate the wavelet coeffi-

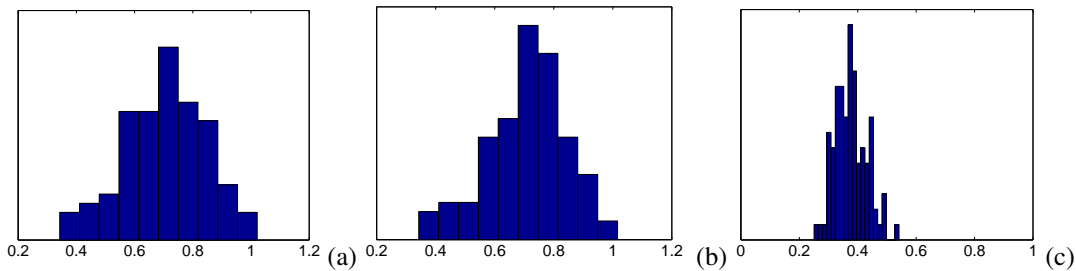

Figure 3: *Improvements afforded by re-weighted $\ell_1$-decoding (a) with known parameters $\boldsymbol{\theta}$ and (b) with learning. (c) The learned $sw\ell_p$ exponent of the GPD distribution with $q = 0.4$ via the RANSAC algorithm.*

cients of natural images that have almost no approximation power of the overall image. Moreover, the learned GGD distribution is dimension dependent, assigns lower probability to the large coefficients that explain the image well, and predicts a mismatched OS of natural images (cf.Fig. 2(b)).

Figure 2(c) compares the magnitude ordered pixel gradients of the images (solid lines) with the expected OS of GGD (dashed line). From the figure, it appears that the natural image pixel gradients lose their compressibility as the image dimensions grow, similar to the GGD, Weibull, gamma, and log-normal distributions. In the figure, the GGD parameters are given $(q, \lambda) = (0.95, 25)$.

### 7.3 Iterative $\ell_1$ Decoding

We repeat the compressible signal recovery experiment in Section 3.2 of [7] to demonstrate the performance of our iterative $\ell_s$ decoder with $s = 1$ in (13). We first randomly sample a signal $\boldsymbol{x} \in \mathbb{R}^N$ ($N = 256$) where the signal coefficients are iid from the GPD distribution with $q = 0.4$ and $\lambda = (N + 1)^{-1/q}$ so that the $E[\bar{x}_{(1)}] \approx 1$. We set $M = 128$ and draw a random $M \times N$ matrix with iid Gaussian entries to obtain $\boldsymbol{y} = \boldsymbol{\Phi}\boldsymbol{x}$. We then decode signals via (13) where maximum iterations is set to 5, with the knowledge of the signal parameters and with learning. During the learning phase, we use $\log(2)$ as the threshold for the RANSAC algorithm. We set the maximum iteration count of RANSAC to 500.

The results of a Monte Carlo run with 100 independent realizations are illustrated in Fig. 3. In Figs. 3(a) and (b), the plots summarize the average improvement over the standard decoder $\Delta_1(\boldsymbol{y})$ via the histograms of $\|\boldsymbol{x} - \widehat{\boldsymbol{x}}_{\{4\}}\|_2/\|\boldsymbol{x} - \Delta_1(\boldsymbol{y})\|_2$, which have mean and standard deviation $(0.7062, 0.1380)$ when we know the parameters of the GPD (a) and $(0.7101, 0.1364)$ when we learn the parameters of the GPD via RANSAC (b). The learned $sw\ell_p$ exponent is summarized by the histogram in Fig. 3(c), which has mean and standard deviation $(0.3757, 0.0539)$. Hence, we conclude that the our alternative learning approach via the RANSAC algorithm is competitive with knowing the actual prior parameters that generated the signal. Moreover, the computational time of learning is insignificant compared to time required by the state-of-the art linear SPGL algorithm [24].

## 8 Conclusions[3]

Compressible priors create a connection between probabilistic and deterministic models for signal compressibility. The bridge between these seemingly two different modeling frameworks turns out to be the concept of order statistics. We demonstrated that when the $p$-parameter of a compressible prior is independent of the ambient dimension $N$, it is possible to have truly logarithmic embedding of its iid signal realizations. Moreover, the learned parameters of such compressible priors are dimension agnostic. In contrast, we showed that when the $p$-parameter depends on $N$, we have many restrictions in signal embedding and recovery as well as in parameter learning. We illustrated that wavelet coefficients of natural images can be well approximated by the generalized Pareto prior, which in turn predicts a disappointing approximation rate for image coding with the naïve sparse model and for CS image recovery from measurements that grow only logarithmically with the image dimension. We motivated many of the existing sparse signal recovery algorithm as instances of a corresponding compressible prior and discussed parameter learning for these priors from dimensionality reduced data. We hope that the iid compressibility view taken in this paper will pave the way for a better understanding of probabilistic non-iid and structured compressibility models.

## Footnotes

[1] To illustrate the issues with the compressibility of GGD, consider the Laplacian distribution (LD: GGD with $q=1$), which is the conventional convex prior for promoting sparsity. Via order statistics, it is possible to show that $\bar{x}_{(i)}\approx\lambda\log\frac{N}{i}$ for $x_i\sim\mathrm{GGD}(1,\lambda)$. Without loss of generality, let us judiciously pick $\lambda=1/\log N$ so that $R=1$. Then, we have $\|\boldsymbol{x}\|_1\approx N-1$ and $\|\boldsymbol{x}-\boldsymbol{x}_K\|_1\approx N-K\log(N/K)-K$. When we only have $K$ terms to capture $(1-\epsilon)$ of the $\ell_1$ energy ($\epsilon\ll 1$) in the signal $\boldsymbol{x}$, we need $K\approx(1-\sqrt{\epsilon})N$.

[2] Here, we assume that the reader is familiar with the discrete wavelet transform and its properties [11].

[3]We thank R. G. Baraniuk, M. Wakin, M. Davies, J. Haupt, and J. P. Slavinksy for useful discussions. Supported by ONR N00014-08-1-1112, DARPA N66001-08-1-2065, ARO W911NF-09-1-0383 grants.

# References

[1] E. J. Candès. Compressive sampling. In *Proc. International Congress of Mathematicians*, volume 3, pages 1433–1452, Madrid, Spain, 2006.

[2] M.E. Tipping. Sparse bayesian learning and the relevance vector machine. *The Journal of Machine Learning Research*, 1:211–244, 2001.

[3] D. P. Wipf and B. D. Rao. Sparse Bayesian learning for basis selection. *IEEE Transactions on Signal Processing*, 52(8):2153–2164, 2004.

[4] T. Blumensath and M.E. Davies. Sampling theorems for signals from the union of linear subspaces. *IEEE Trans. Info. Theory*, 2009.

[5] A. Cohen, W. Dahmen, and R. DeVore. Compressed sensing and best k-term approximation. *American Mathematical Society*, 22(1):211–231, 2009.

[6] I. F. Gorodnitsky, J. S. George, and B. D. Rao. Neuromagnetic source imaging with FOCUSS: a recursive weighted minimum norm algorithm. *Electroenceph. and Clin. Neurophys.*, 95(4):231–251, 1995.

[7] E. J. Candès, M. B. Wakin, and S. P. Boyd. Enhancing sparsity by reweighted $\ell_1$ minimization. *Journal of Fourier Analysis and Applications*, 14(5):877–905, 2008.

[8] D. P. Wipf and S. Nagarajan. Iterative reweighted $\ell_1$ and $\ell_2$ methods for finding sparse solutions. In *SPARS09*, Rennes, France, 2009.

[9] S. S. Chen, D. L. Donoho, and M. A. Saunders. Atomic decomposition by basis pursuit. *SIAM review*, pages 129–159, 2001.

[10] H.A. David and H.N. Nagaraja. *Order Statistics*. Wiley-Interscience, 2004.

[11] S. Mallat. *A Wavelet Tour of Signal Processing*. Academic Press, 1999.

[12] H. Choi and R. G. Baraniuk. Wavelet statistical models and Besov spaces. *Lecture Notes in Statistics*, pages 9–30, 2003.

[13] T. K. Nayak. Multivariate Lomax distribution: properties and usefulness in reliability theory. *Journal of Applied Probability*, pages 170–177, 1987.

[14] V. Cevher. Compressible priors. *IEEE Trans. on Information Theory, in preparation,* 2010.

[15] R. Tibshirani. Regression shrinkage and selection via the lasso. *Journal of the Royal Statistical Society*, pages 267–288, 1996.

[16] E. T. Hale, W. Yin, and Y. Zhang. Fixed-point continuation for $\ell_1$-minimization: Methodology and convergence. *SIAM Journal on Optimization*, 19:1107, 2008.

[17] P. J. Garrigues. *Sparse Coding Models of Natural Images: Algorithms for Efficient Inference and Learning of Higher-Order Structure*. PhD thesis, EECS Department, University of California, Berkeley, May 2009.

[18] M. J. Wainwright and E. P. Simoncelli. Scale mixtures of Gaussians and the statistics of natural images. In *NIPS*, 2000.

[19] D. Wipf and S. Nagarajan. A new view of automatic relevance determination. In *NIPS*, volume 20, 2008.

[20] I. Daubechies, R. DeVore, M. Fornasier, and S. Gunturk. Iteratively re-weighted least squares minimization for sparse recovery. *Commun. Pure Appl. Math*, 2009.

[21] R. Chartrand and W. Yin. Iteratively reweighted algorithms for compressive sensing. In *ICASSP*, pages 3869–3872, 2008.

[22] M.A. Fischler and R.C. Bolles. Random sample consensus: a paradigm for model fitting with applications to image analysis and automated cartography. *Communications of the ACM*, 24(6):381–395, 1981.

[23] E. J. Candes and D. L. Donoho. Curvelets and curvilinear integrals. *Journal of Approximation Theory*, 113(1):59–90, 2001.

[24] E. van den Berg and M. P. Friedlander. Probing the Pareto frontier for basis pursuit solutions. *SIAM Journal on Scientific Computing*, 31(2):890–912, 2008.

